# Call-based Fraud Detection in Mobile Communication Networks using a Hierarchical Regime-Switching Model

**Jaakko Hollmén**
Helsinki University of Technology
Lab. of Computer and Information Science
P.O. Box 5400, 02015 HUT, Finland
Jaakko.Hollmen@hut.fi

**Volker Tresp**
Siemens AG, Corporate Technology
Dept. Information and Communications
81730 Munich, Germany
Volker.Tresp@mchp.siemens.de

## Abstract

Fraud causes substantial losses to telecommunication carriers. Detection systems which automatically detect illegal use of the network can be used to alleviate the problem. Previous approaches worked on features derived from the call patterns of individual users. In this paper we present a call-based detection system based on a hierarchical regime-switching model. The detection problem is formulated as an inference problem on the regime probabilities. Inference is implemented by applying the junction tree algorithm to the underlying graphical model. The dynamics are learned from data using the EM algorithm and subsequent discriminative training. The methods are assessed using fraud data from a real mobile communication network.

## 1 INTRODUCTION

Fraud is costly to a network carrier both in terms of lost income and wasted capacity. It has been estimated that the telecommunication industry looses approximately 2-5% of its total revenue to fraud. The true losses are expected to be even higher since telecommunication companies are reluctant to admit fraud in their systems. A fraudulent attack causes lots of inconveniences to the victimized subscriber which might motivate the subscriber to switch to a competing carrier. Furthermore, potential new customers would be very reluctant to switch to a carrier which is troubled with fraud.

Mobile communication networks —which are the focus of this work— are particularly appealing to fraudsters as the calling from the mobile terminal is not bound to a physical place and a subscription is easy to get. This provides means for an illegal high-profit business requiring minimal investment and relatively low risk of getting caught. Fraud is

usually initiated by a mobile phone theft, by cloning the mobile phone card or by acquiring a subscription with false identification. After intrusion the subscription can be used for gaining free services either for the intruder himself or for his illegal customers in form of call-selling. In the latter case, the fraudster sells calls to customers for reduced rates.

The earliest means of detecting fraud were to register overlapping calls originating from one subscription, evidencing card cloning. While this procedure efficiently detects cloning, it misses a large share of other fraud cases. A more advanced system is a velocity trap which detects card cloning by using an upper speed limit at which a mobile phone user can travel. Subsequent calls from distant places provide evidence for card cloning. Although a velocity trap is a powerful method of detecting card cloning, it is ineffective against other types of fraud. Therefore there is great interest in detection systems which detect fraud based on an analysis of behavioral patterns (Barson *et al.*, 1996, Burge *et al.*, 1997, Fawcett and Provost, 1997, Taniguchi *et al.*, 1998).

In an absolute analysis, a user is classified as a fraudster based on features derived from daily statistics summarizing the call pattern such as the average number of calls. In a differential analysis, the detection is based on measures describing the changes in those features capturing the transition from a normal use to fraud. Both approaches have the problem of finding efficient feature representations describing normal and fraudulent behavior. As they usually derive features as summary statistics over one day, they are plagued with a latency time of up to a day to detect fraudulent behavior. The resulting delay in detection can already lead to unacceptable losses and can be exploited by the fraudster. For these reasons real-time fraud detection is seen to be the most important development in fraud detection (Pequeno, 1997).

In this paper we present a real-time fraud detection system which is based on a stochastic generative model. In the generative model we assume a variable *victimized* which indicates if the account has been victimized by a fraudster and a second variable *fraud* which indicates if the fraudster is currently performing fraud. Both variables are hidden. Furthermore, we have an observed variable *call* which indicates if a call being is performed or not. The transition probabilities from no-call to call and from call to no-call are dependent on the state of the variable *fraud*. Overall, we obtain a regime-switching time-series model as described by Hamilton (1994), with the modifications that first, the variables in the time series are not continuous but binary and second, the switching variable has a hierarchical structure. The benefit of the hierarchical structure is that it allows us to model the time-series at different time scales. At the lowest hierarchical level, we model the dynamical behavior of the individual calls, at the next level the transition from normal behavior to fraudulent behavior and at the highest level the transition to being victimized. To be able to model a time-series at different temporal resolutions was also the reason for introducing a hierarchy into a hidden Markov model for Jordan, Ghahramani and Saul (1997). Fortunately, our hidden variables have only a small number of states such that we do not have to work with the approximation techniques those authors have introduced.

Section 2 introduces our hierarchical regime-switching fraud model. The detection problem is formulated as an inference problem on the regime probabilities based on subscriber data. We derive iterative algorithms for estimating the hidden variables *fraud* and *victimized* based on past and present data (filtering) or based on the complete set of observed data (smoothing). We present EM learning rules for learning the parameters in the model using observed data. We develop a gradient based approach for fine tuning the emission probabilities in the non-fraud state to enhance the discrimination capability of the model. In Section 3 we present experimental results. We show that a system which is fine-tuned on real data can be used for detecting fraudulent behavior on-line based on the call patterns. In Section 4 we present conclusions and discuss further applications and extensions of our fraud model.

## 2 THE HIERARCHICAL REGIME-SWITCHING FRAUD MODEL

### 2.1 THE GENERATIVE MODEL

The hierarchical regime-switching model consists of three variables which evolve in time stochastically according to first-order Markov chains. The first binary variable $v_t$ (victimized) is equal to one if the account is currently being victimized by a fraudster and zero otherwise. The states of this variable evolve according to the state transition probabilities $p_{ij}^v = P(v_t = i|v_{t-1} = j); i, j = 0, 1$. The second binary variable $s_t$ (fraud) is equal to one if the fraudster currently performs fraud and is equal to zero if the fraudster is inactive. The change between actively performing fraud and intermittent silence is typical for a victimized account as is apparent from Figure 3. Note that this transient bursty behavior of a victimized account would be difficult to capture with a pure feature based approach. The states of this variable evolve following the state transition probabilities $p_{ijk}^s = P(s_t = i|v_t = j, s_{t-1} = k, ); i, j, k = 0, 1$. Finally, the binary variable $y_t$ (call) is equal to one if the mobile phone is being used and zero otherwise with state transition matrix $p_{ijk}^y = P(y_t = i|s_t = j, y_{t-1} = k); i, j, k = 0, 1$. Note that this corresponds to the assumption of exponentially distributed call duration. Although not quite realistic, this is the general assumption in telecommunications. Typically, both the frequency of calls and the lengths of the calls are increased when fraud is executed. The joint probability of the time series up to time $T$ is then

$$P(V_T, S_T, Y_T) = P(v_0, s_0, y_0) \prod_{t=1}^{T} P(v_t|v_{t-1}) \prod_{t=1}^{T} P(s_t|v_t, s_{t-1}) \prod_{t=1}^{T} P(y_t|s_t, y_{t-1}) \quad (1)$$

where in the experiments we used a sampling time of one minute. Furthermore, $V_T = \{v_0, \ldots, v_T\}$, $S_T = \{s_0, \ldots, s_T\}$, $Y_T = \{y_0, \ldots, y_T\}$ and $P(v_0, s_0, y_0)$ is the prior distribution of the initial states.

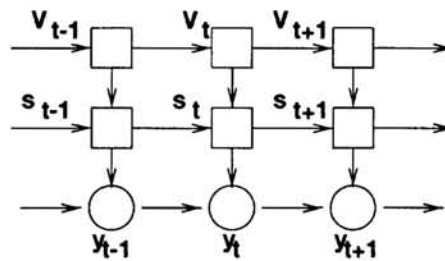

Figure 1: Dependency graph of the hierarchical regime-switching fraud model. The square boxes denote hidden variables and the circles observed variables. The hidden variable $v_t$ on the top describes whether the subscriber account is victimized by fraud. The hidden variable $s_t$ indicates if fraud is currently being executed. The state of $s_t$ determines the statistics of the variable call $y_t$.

### 2.2 INFERENCE: FILTERING AND SMOOTHING

When using the fraud detection system, we are interested to estimate the probability that an account is victimized or that fraud is currently occurring based on the call patterns up to the current point in time (filtering). We can calculate the probabilities of the states of the hidden variables by applying the following equations recursively with $t = 1, \ldots, T$.

$$P(v_t = i, s_{t-1} = k | Y_{t-1}) = \sum_l p_{il}^v P(v_{t-1} = l, s_{t-1} = k | Y_{t-1})$$

$$P(v_t = i, s_t = j | Y_{t-1}) = \sum_k p_{jik}^s P(v_t = i, s_{t-1} = k | Y_{t-1})$$

$$P(v_t = i, s_t = j | Y_t) = c \cdot p_{y_t j y_{t-1}}^y P(v_t = i, s_t = j | Y_{t-1})$$

where $c$ is a scaling factor. These equations can be derived from the junction tree algorithm for the Bayesian networks (Jensen, 1996). We obtain the probability of victimization and fraud by simple marginalization

$$P(v_t = i | Y_t) = \sum_j P(v_t = i, s_t = j | Y_t) \; ; \; P(s_t = j | Y_t) = \sum_i P(v_t = i, s_t = j | Y_t).$$

In some cases —in particular for the EM learning rules in the next section— we might be interested in estimating the probabilities of the hidden states at some time in the past (smoothing). In this case we can use a variation of the smoothing equations described in Hamilton (1994) and Kim (1994). After performing the forward recursion, we can calculate the probability of the hidden states at time $t'$ given data up to time $T > t'$ iterating the following equations with $t = T, T - 1, \ldots, 1$.

$$P(v_{t+1} = k, s_t = j | Y_T) = \sum_l \frac{P(v_{t+1} = k, s_{t+1} = l | Y_T)}{P(v_{t+1} = k, s_{t+1} = l | Y_t)} P(v_{t+1} = k, s_t = j | Y_t) p_{lkj}^s$$

$$P(v_t = i, s_t = j | Y_T) = \sum_k \frac{P(v_{t+1} = k, s_t = j | Y_T)}{P(v_{t+1} = k, s_t = j | Y_t)} P(v_t = i, s_t = j | Y_t) p_{ki}^v$$

## 2.3   EM LEARNING RULES

Parameter estimation in the regime-switching model is conveniently formulated as an incomplete data problem, which can be solved using the EM algorithm (Hamilton, 1994). Each iteration of the EM algorithm is guaranteed to increase the value of the marginal log-likelihood function until a fixed point is reached. This fixed point is a local optimum of the marginal log-likelihood function.

In the M-step the model parameters are optimized using the estimates of the hidden states using the current parameter estimates. Let $\theta = \{p_{ij}^v, p_{ijk}^s, p_{ikj}^y\}$ denote the current parameter estimates. The new estimates are obtained using

$$p_{ij}^v = \frac{\sum_{t=1}^T P(v_t = i, v_{t-1} = j | Y_T; \theta)}{\sum_{t=1}^T P(v_{t-1} = j | Y_T; \theta)}$$

$$p_{ijk}^s = \frac{\sum_{t=1}^T P(s_t = i, v_t = j, s_{t-1} = k | Y_T; \theta)}{\sum_{t=1}^T P(v_t = j, s_{t-1} = k | Y_T; \theta)}$$

$$p_{ikj}^y = \frac{\sum_{t=1, if\ y_t = i\ and\ y_{t-1}=j}^T P(s_{t-1} = k | Y_T; \theta)}{\sum_{t=1,\ if\ y_{t-1}=j}^T P(s_{t-1} = k | Y_T; \theta)}$$

The E-step determines the probabilities on the right sides of the equations using the current parameter estimates. These can be determined using the smoothing equations from the previous section directly by marginalizing

$$P(v_t = k, s_t = l, v_{t+1} = i, s_{t+1} = j | Y_T)$$

$$= P(v_{t+1} = i, s_{t+1} = j | Y_T) \frac{p_{ik}^v p_{jkl}^s P(v_t = k, s_t = l | Y_t)}{P(v_{t+1} = i, s_{t+1} = j | Y_t)}$$

where the terms on the right side are obtained from the equations in the last Section.

## 2.4 DISCRIMINATIVE TRAINING

In our data setting, it is not known when the fraudulent accounts were victimized by fraud. This is why we use the EM algorithm to learn the two regimes from data in an incomplete data setting. We know, however, which accounts were victimized by fraud. After EM learning the discrimination ability of the model was not satisfactory. We therefore used the labeled sequences to improve the model. The reason for the poor performance was credited to unsuitable call emission probabilities in the normal state. We therefore minimize the error function $E = \sum_i \left( \max_t P(v_t^{(i)} | Y_t^{(i)}) - t^{(i)} \right)^2$ with regard to the parameter $p_{i=0,j=0,k=0}^y$, where the $t^{(i)} = \{0, 1\}$ is the label for the sequence $i$. The error function was minimized with Quasi-Newton procedure with numerical differentiation.

# 3 EXPERIMENTS

To test our approach we used a data set consisting of 600 accounts which were not affected by fraud and 304 accounts which were affected by fraud. The time period for non-fraud and fraud accounts were 49 and 92 days, respectively. We divided the data equally into training data and test data. From the non-fraud data we estimated the parameters describing the normal calling behavior, i.e. $p_{i,j=0,k}^y$. Next, we fixed the probability that an account is victimized from one time step to the next to $p_{i=1,j=0}^v = 10^{-5}$ and the probability that a victimized account becomes de-victimized as $p_{i=0,j=1}^v = 5 \times 10^{-4}$. Leaving those parameters fixed the remaining parameters were trained using the fraudulent accounts and the EM algorithm described in Section 2. We had to do unsupervised training since it was known by velocity check that the accounts were affected but it was not clear when the intrusion occurred. After unsupervised training, we further enhanced the discrimination capability of the system which helped us reduce the amount of false alarms. The final model parameters can be found in the Appendix.

After training, the system was tested using the test data. Unfortunately, it is not known when the accounts were attacked by fraud, but only on per-account basis if an account was at some point a victim of fraud. Therefore, we declare an account to be victimized if the victimized variable at some point exceeds the threshold. Also, it is interesting to study the results shown in Figure 3. We show data and posterior time-evolving probabilities for an account which is known to be victimized. From the call pattern it is obvious that there are periods of suspiciously high traffic at which the probability of victimization is recognized to be very high. We also see that the variable fraud $s_t$ follows the bursty behavior of the fraudulent behavior correctly. Note, that for smoothing which is important both for a retrospective analysis of call data and for learning, we achieve smoother curves for the victimized variable.

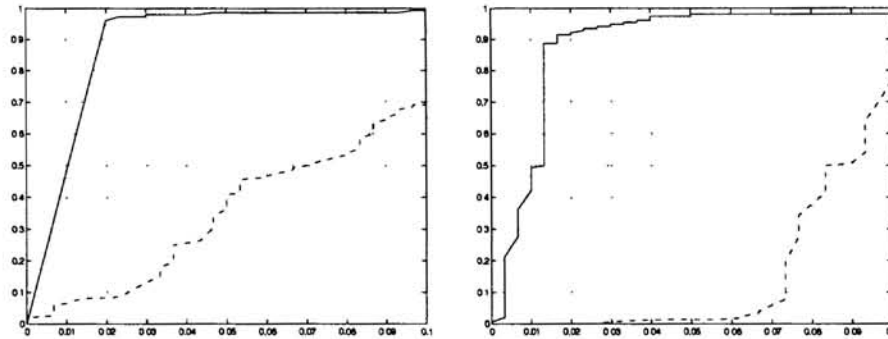

Figure 2: The Receiver Operating Characteristic (ROC) curves are shown for on-line detection (left figure) and for retrospective classification (right figure). In the figures, detection probability is plotted against the false alarm probability. The dash-dotted lines are results before, the solid lines after discriminative training. We can see that the discriminative training improves the model considerably.

After EM training and discriminative training, we tested the model both in on-line detection mode (filtering) and in retrospective classification (smoothing) with smoothed probabilities. The detection results are shown in Figure 2. With a fixed false alarm probability of 0.003, the detection probabilities for the training set were found to be 0.974 and 0.934 using on-line detection mode and with smoothed probabilities, respectively. With a testing set and a fixed false alarm probability of 0.020, we obtain the detection probabilities of 0.928 and 0.921, for the on-line detection and for retrospective classification, respectively.

## 4   CONCLUSIONS

We presented a call-based on-line fraud detection system which is based on a hierarchical regime-switching generative model. The inference rules are obtained from the junction tree algorithm for the underlying graphical model. The model is trained using the EM algorithm in an incomplete data setting and is further refined with gradient-based discriminative training, which considerably improves the results.

A few extensions are in the process of being implemented. First of all, it makes sense to use more than one fraud model for the different fraud scenarios and several user models to account for different user profiles. For these more complex models we might have to rely on approximations techniques such as the ones introduced by Jordan, Ghahramani and Saul (1997).

### Appendix

The model parameters after EM training and discriminative training. Note that entering the fraud state without first entering the victimized state is impossible.

$$p^y_{i,j=0,k} = \begin{pmatrix} 0.9559 & 0.0441 \\ 0.3533 & 0.6467 \end{pmatrix} \quad p^y_{i,j=1,k} = \begin{pmatrix} 0.9292 & 0.0708 \\ 0.0570 & 0.9430 \end{pmatrix}$$

$$p^s_{i,j=0,k} = \begin{pmatrix} 1.0000 & 0.0000 \\ 0.0000 & 1.0000 \end{pmatrix} \quad p^s_{i,j=1,k} = \begin{pmatrix} 0.9979 & 0.0021 \\ 0.0086 & 0.9914 \end{pmatrix}$$

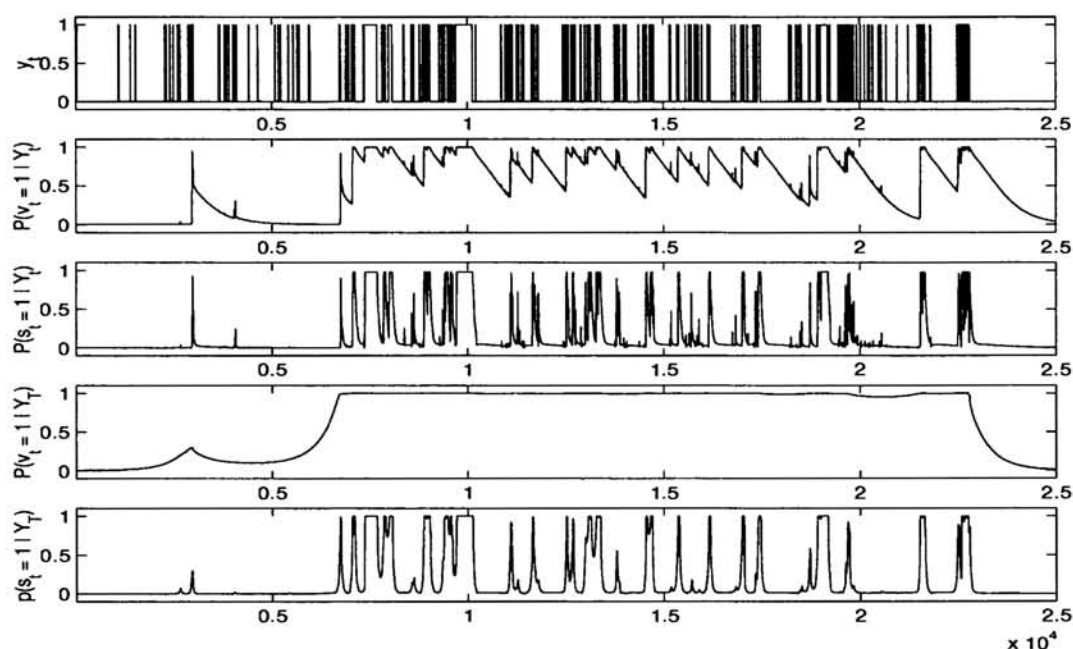

Figure 3: The first line shows the calling data $y_t$ from a victimized account. The second and third lines show the states of the victimized and fraud variables, respectively. Both are calculated with the filtering equations. The fourth and fifth lines show the same variables using the smoothing equations. The displayed time window period is seventeen days.

## References

Barson P., Field, S., Davey, N., McAskie, G., and Frank, R. (1996). The Detection of Fraud in Mobile Phone Networks. *Neural Network World,* Vol. 6, No. 4.

Bengio, Y. (1996). Markovian Models for Sequential Data. *Technical Report # 1049, Université de Montreal.*

Burge, P., Shawe-Taylor J., Moreau Y., Verrelst, H., Störmann C. and Gosset, P. (1997). BRUTUS - A Hybrid Detection Tool. *Proc. of ACTS Mobile Telecommunications Summit, Aalborg, Denmark.*

Fawcett, T. and Provost, F. (1997). Adaptive Fraud Detection. *Journal of Data Mining and Knowledge Discovery,* , Vol. 1, No. 3, pp. 1-28.

Hamilton, J. D. (1994). *Time Series Analysis.* Princeton University Press.

Jensen, Finn V. (1996). *Introduction to Bayesian Networks.* UCL Press.

Jordan, M. I, Ghahramani, Z. and Saul, L. K. (1997). Hidden Markov Decision Trees, in *Advances in Neural Information Processing Systems: Proceedings of the 1996 Conference (NIPS'9),* MIT-Press, pp. 501-507.

Kim, C.-J. (1994). Dynamical linear models with Markov-switching. *Journal of Econometrics,* Vol. 60, pp. 1-22.

Pequeno, K. A.(1997). Real-Time fraud detection: Telecom's next big step. *Telecommunications (America Edition),* Vol. 31, No. 5, pp. 59-60.

Taniguchi, M., Haft, M., Hollmén, J. and Tresp, V. (1998). Fraud detection in communications networks using neural and probabilistic methods. *Proceedings of the 1998 IEEE Int. Conf. in Acoustics, Speech and Signal Processing (ICASSP'98),* Vol. 2, pp. 1241-1244.
